# Discovering high order features with mean field modules

**Conrad C. Galland and Geoffrey E. Hinton**
Physics Dept. and Computer Science Dept.
University of Toronto
Toronto, Canada
M5S 1A4

## ABSTRACT

A new form of the deterministic Boltzmann machine (DBM) learning procedure is presented which can efficiently train network modules to discriminate between input vectors according to some criterion. The new technique directly utilizes the free energy of these "mean field modules" to represent the probability that the criterion is met, the free energy being readily manipulated by the learning procedure. Although conventional deterministic Boltzmann learning fails to extract the higher order feature of shift at a network bottleneck, combining the new mean field modules with the mutual information objective function rapidly produces modules that perfectly extract this important higher order feature without direct external supervision.

## 1  INTRODUCTION

The Boltzmann machine learning procedure (Hinton and Sejnowski, 1986) can be made much more efficient by using a mean field approximation in which stochastic binary units are replaced by deterministic real-valued units (Peterson and Anderson, 1987). Deterministic Boltzmann learning can be used for "multicompletion" tasks in which the subsets of the units that are treated as input or output are varied from trial to trial (Peterson and Hartman, 1988). In this respect it resembles other learning procedures that also involve settling to a stable state (Pineda, 1987). Using the multicompletion paradigm, it should be possible to force a network to explicitly extract important higher order features of an ensemble of training vectors by forcing the network to pass the information required for correct completions through a narrow bottleneck. In back-propagation networks with two or three hidden layers, the use of bottlenecks sometimes allows the learning to explictly discover important

underlying features (Hinton, 1986) and our original aim was to demonstrate that the same idea could be used effectively in a DBM with three hidden layers. The initial simulations using conventional techniques were not successful, but when we combined a new type of DBM learning with a new objective function, the resulting network extracted the crucial higher order features rapidly and perfectly.

## 2    THE MULTI-COMPLETION TASK

Figure 1 shows a network in which the input vector is divided into 4 parts. A1 is a random binary vector. A2 is generated by shifting A1 either to the right or to the left by one "pixel", using wraparound. B1 is also a random binary vector, and B2 is generated from B1 by using the same shift as was used to generate A2 from A1. This means that any three of A1, A2, B1, B2 uniquely specify the fourth (we filter out the ambiguous cases where this is not true). To perform correct completion, the network must explicitly represent the shift in the single unit that connects its two halves. Shift is a second order property that cannot be extracted without hidden units.

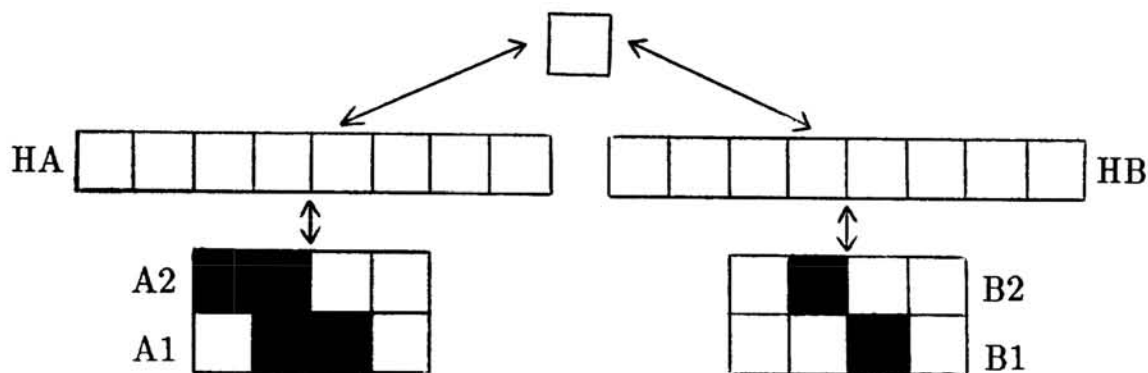

Figure 1.

## 3    SIMULATIONS USING STANDARD DETERMINISTIC BOLTZMANN LEARNING

The following discussion assumes familiarity with the deterministic Boltzmann learning procedure, details of which can be obtained from Hinton (1989). During the positive phase of learning, each of the 288 possible sets of shift matched four-bit vectors were clamped onto inputs A1, A2 and B1, B2, while in the negative phase, one of the four was allowed to settle unclamped. The weights were changed after each training case using the on-line version of the DBM learning procedure. The choice of which input not to clamp changed systematically throughout the learning process so that each was left unclamped equally often. This technique, although successful in problems with only one hidden layer, could not train the network to correctly perform the multicompletion task where any of the four input layers would settle to the correct state when the other three were clamped. As a result, the single

central unit failed to extract shift. In general, the DBM learning procedure, like its stochastic predecessor, seems to have difficulty learning tasks in multi-hidden layer nets. This failure led to the development of the new procedure which, in one form, manages to correctly extract shift without the need for many hidden layers or direct external supervision.

# 4   A NEW LEARNING PROCEDURE FOR MEAN FIELD MODULES

A DBM with unit states in the range $[-1, 1]$ has free energy

$$F = -\sum_{i<j} y_i y_j w_{ij} + T \sum_i \left[ \frac{(1+y_i)}{2} \log \frac{(1+y_i)}{2} + \frac{(1-y_i)}{2} \log \frac{(1-y_i)}{2} \right] \quad (1)$$

The DBM settles to a free energy minimum, $F^*$, at a non-zero temperature, where the states of the units are given by

$$y_i = \tanh(\frac{1}{T} \sum_j y_j w_{ij}) \quad (2)$$

At the minimum, the derivative of $F^*$ with respect to a particular weight (assuming $T = 1$) is given by (Hinton, 1989)

$$\frac{\cdot \partial F^*}{\partial w_{ij}} = -y_i y_j \quad (3)$$

Suppose that we want a network module to discriminate between input vectors that "fit" some criterion and input vectors that don't. Instead of using a net with an output unit that indicates the degree of fit, we could view the negative of the mean field free energy of the whole module as a measure of how happy it is with the clamped input vector. From this standpoint, we can define the probability that input vector $\alpha$ fits the criterion as

$$p_\alpha = \frac{1}{(1 + e^{F^*_\alpha})} \quad (4)$$

where $F^*_\alpha$ is the equilibrium free energy of the module with vector $\alpha$ clamped on the inputs.

Supervised training can be performed by using the cross-entropy error function (Hinton, 1987):

$$C = -\sum_{i=\alpha}^{N_+} \log(p_\alpha) - \sum_{j=\beta}^{N_-} \log(1 - p_\beta) \quad (5)$$

where the first sum is over the $N_+$ input cases that fit the criterion, and the second is over the $N_-$ cases that don't. The cross-entropy expression is used to specify error

derivatives for $p_\alpha$ and hence for $F_\alpha^*$. Error derivatives for each weight can then be obtained by using equation (3), and the module is trained by gradient descent to have high free energy for the "negative" training cases and low free energy for the "positive" cases.

Thus, for each positive case

$$
\begin{aligned}
-\frac{\partial \log(p_\alpha)}{\partial w_{ij}} &= \frac{1}{1 + e^{F_\alpha^*}} \, e^{F_\alpha^*} \, \frac{\partial F_\alpha^*}{\partial w_{ij}} \\
&= \frac{1}{1 + e^{-F_\alpha^*}} \, (-y_i y_j)
\end{aligned}
$$

For each negative case,

$$
\begin{aligned}
-\frac{\partial \log(1 - p_\beta)}{\partial w_{ij}} &= -\frac{1}{1 + e^{-F_\beta^*}} \, e^{-F_\beta^*} \, \frac{\partial F_\beta^*}{\partial w_{ij}} \\
&= \frac{1}{1 + e^{F_\beta^*}} \, (y_i y_j)
\end{aligned}
$$

To test the new procedure, we trained a shift detecting module, composed of the the input units A1 and A2 and the hidden units HA from figure 1, to have low free energy for all and only the right shifts. Each weight was changed in an on-line fashion according to

$$
\Delta w_{ij} = \epsilon \, \frac{1}{1 + e^{-F_\alpha^*}} \, y_i y_j
$$

for each right shifted case, and

$$
\Delta w_{ij} = -\epsilon \, \frac{1}{1 + e^{F_\beta^*}} \, y_i y_j
$$

for each left shifted case. Only 10 sweeps through the 24 possible training cases were required to successfully train the module to detect shift. The training was particularly easy because the hidden units only receive connections from the input units which are always clamped, so the network settles to a free energy minimum in one iteration. Details of the simulations are given in Galland and Hinton (1990).

## 5    MAXIMIZING MUTUAL INFORMATION BETWEEN MEAN FIELD MODULES

At first sight, the new learning procedure is inherently supervised, so how can it be used to *discover* that shift is an important underlying feature? One method

is to use two modules that each supervise the other. The most obvious way of implementing this idea quickly creates modules that always agree because they are always "on". If, however, we try to maximize the mutual information between the stochastic binary variables represented by the free energies of the modules, there is a strong pressure for each binary variable to have high entropy across cases because the mutual information between binary variables A and B is:

$$I(A;B) = H_A + H_B - H_{AB} \tag{6}$$

where $H_{AB}$ is the entropy of the joint distribution of A and B over the training cases, and $H_A$ and $H_B$ are the entropies of the individual distributions.

Consider two mean field modules with associated stochastic binary variables $A,B \in \{0,1\}$. For a given case $\alpha$,

$$p(A^\alpha = 1) = \frac{1}{1 + e^{F_{A,\alpha}^*}} \tag{7}$$

where $F_{A,\alpha}^*$ is the free energy of the $A$ module with the training case $\alpha$ clamped on the input.

We can compute the probability that the $A$ module is on or off by averaging over the input sample distribution, with $P^\alpha$ being the prior probability of an input case $\alpha$:

$$p(A=1) = \sum_\alpha P^\alpha p(A^\alpha = 1)$$

$$p(A=0) = 1 - p(A=1)$$

Similarly, we can compute the four possible values in the joint probability distribution of $A$ and $B$:

$$p(A=1, B=1) = \sum_\alpha P^\alpha p(A^\alpha = 1) p(B^\alpha = 1)$$

$$p(A=0, B=1) = p(B=1) - p(A=1, B=1)$$

$$p(A=1, B=0) = p(A=1) - p(A=1, B=1)$$

$$p(A=0, B=0) = 1 - p(B=1) - p(A=1) + p(A=1, B=1)$$

Using equation (3), the partial derivatives of the various individual and joint probability functions with respect to a weight $w_{ik}$ in the $A$ module are readily calculated.

$$\frac{\partial p(A=1)}{\partial w_{ik}} = \sum_\alpha P^\alpha \frac{\partial p(A^\alpha = 1)}{\partial w_{ik}}$$

$$= \sum_\alpha P^\alpha \left( p(A^\alpha = 1) - 1 \right) p(A^\alpha = 1)(y_i y_k) \tag{8}$$

$$\frac{\partial p(A=1, B=1)}{\partial w_{ik}} = \sum_{\alpha} P^{\alpha} \frac{\partial p(A^{\alpha}=1)}{\partial w_{ik}} p(B^{\alpha}=1) \qquad (9)$$

The entropy of the stochastic binary variable $A$ is

$$H_A = - <\log p(A)> = - \sum_{a=0,1} p(A=a) \log p(A=a)$$

The entropy of the joint distribution is given by

$$
\begin{aligned}
H_{AB} &= - <\log p(A, B)> \\
&= - \sum_{a,b} p(A=a, B=b) \log p(A=a, B=b)
\end{aligned}
$$

The partial derivative of $I(A; B)$ with respect to a single weight $w_{ik}$ in the $A$ module can now be computed; since $H_B$ does not depend on $w_{ik}$, we need only differentiate $H_A$ and $H_{AB}$. As shown in Galland and Hinton (1990), the derivative is given by

$$
\begin{aligned}
\frac{\partial I(A; B)}{\partial w_{ik}} &= \frac{\partial H_A}{\partial w_{ik}} - \frac{\partial H_{AB}}{\partial w_{ik}} \\
&= \sum_{\alpha} P^{\alpha} \left( p(A^{\alpha}=1) - 1 \right) p(A^{\alpha}=1)(y_i y_k) \left[ \log \frac{p(A=1)}{p(A=0)} \right. \\
&\quad \left. - p(B^{\alpha}=1) \log \frac{p(A=1, B=1)}{p(A=0, B=1)} - p(B^{\alpha}=0) \log \frac{p(A=1, B=0)}{p(A=0, B=0)} \right]
\end{aligned}
$$

The above derivation is drawn from Becker and Hinton (1989) who show that mutual information can be used as a learning signal in back-propagation nets. We can now perform gradient ascent in $I(A; B)$ for each weight in both modules using a two-pass procedure, the probabilities across cases being accumulated in the first pass.

This approach was applied to a system of two mean field modules (the left and right halves of figure 1 without the connecting central unit) to detect shift. As in the multi-completion task, random binary vectors were clamped onto inputs A1, A2 and B1, B2 related only by shift. Hence, the only way the two modules can provide mutual information to each other is by representing the shift. Maximizing the mutual information between them created perfect shift detecting modules in only 10 two-pass sweeps through the 288 training cases. That is, after training, each module was found to have low free energy for either left or right shifts, and high free energy for the other. Details of the simulations are again given in Galland and Hinton (1990).

# 6  SUMMARY

Standard deterministic Boltzmann learning failed to extract high order features in a network bottleneck. We then explored a variant of DBM learning in which the free energy of a module represents a stochastic binary variable. This variant can efficiently discover that shift is an important feature without using external supervision, provided we use an architecture and an objective function that are designed to extract higher order features which are invariant across space.

## Acknowledgements

We would like to thank Sue Becker for many helpful comments. This research was supported by grants from the Ontario Information Technology Research Center and the National Science and Engineering Research Council of Canada. Geoffrey Hinton is a fellow of the Canadian Institute for Advanced Research.

## References

Becker, S. and Hinton, G. E. (1989). Spatial coherence as an internal teacher for a neural network. Technical Report CRG-TR-89-7, University of Toronto.

Galland, C. C. and Hinton, G. E. (1990). Experiments on discovering high order features with mean field modules. University of Toronto Connectionist Research Group Technical Report, forthcoming.

Hinton, G. E. (1986) Learning distributed representations of concepts. *Proceedings of the Eighth Annual Conference of the Cognitive Science Society*, Amherst, Mass.

Hinton, G. E. (1987) Connectionist learning procedures. Technical Report CMU-CS-87-115, Carnegie Mellon University.

Hinton, G. E. (1989) Deterministic Boltzmann learning performs steepest descent in weight-space. *Neural Computation*, **1**.

Hinton, G. E. and Sejnowski, T. J. (1986) Learning and relearning in Boltzmann machines. In Rumelhart, D. E., McClelland, J. L., and the PDP group, *Parallel Distributed Processing: Explorations in the Microstructure of Cognition. Volume 1: Foundations*, MIT Press, Cambridge, MA.

Hopfield, J. J. (1984) Neurons with graded response have collective computational properties like those of two-state neurons. *Proceedings of the National Academy of Sciences U.S.A.*, **81**, 3088–3092.

Peterson, C. and Anderson, J. R. (1987) A mean field theory learning algorithm for neural networks. *Complex Systems*, **1**, 995–1019.

Peterson, C. and Hartman, E. (1988) Explorations of the mean field theory learning algorithm. Technical Report ACA-ST/HI-065-88, Microelectronics and Computer Technology Corporation, Austin, TX.

Pineda, F. J. (1987) Generalization of backpropagation to recurrent neural networks. *Phys. Rev. Lett.*, **18**, 2229–2232.